# Grammatical Bigrams

**Mark A. Paskin**
Computer Science Division
University of California, Berkeley
Berkeley, CA 94720
*paskin@cs.berkeley.edu*

## Abstract

Unsupervised learning algorithms have been derived for several statistical models of English grammar, but their computational complexity makes applying them to large data sets intractable. This paper presents a probabilistic model of English grammar that is much simpler than conventional models, but which admits an efficient EM training algorithm. The model is based upon *grammatical bigrams*, i.e., syntactic relationships between pairs of words. We present the results of experiments that quantify the representational adequacy of the grammatical bigram model, its ability to generalize from labelled data, and its ability to induce syntactic structure from large amounts of raw text.

## 1 Introduction

One of the most significant challenges in learning grammars from raw text is keeping the computational complexity manageable. For example, the EM algorithm for the unsupervised training of Probabilistic Context-Free Grammars—known as the Inside-Outside algorithm—has been found in practice to be "computationally intractable for realistic problems" [1]. Unsupervised learning algorithms have been designed for other grammar models (e.g., [2, 3]). However, to the best of our knowledge, no large-scale experiments have been carried out to test the efficacy of these algorithms; the most likely reason is that their computational complexity, like that of the Inside-Outside algorithm, is impractical.

One way to improve the complexity of inference and learning in statistical models is to introduce independence assumptions; however, doing so increases the model's bias. It is natural to wonder how a simpler grammar model (that can be trained efficiently from raw text) would compare with conventional models (which make fewer independence assumptions, but which must be trained from labelled data). Such a model would be a useful tool in domains where partial accuracy is valuable and large amounts of unlabelled data are available (e.g., Information Retrieval, Information Extraction, etc.).

In this paper, we present a probabilistic model of syntax that is based upon *grammatical bigrams*, i.e., syntactic relationships between pairs of words. We show how this model results from introducing independence assumptions into more conven-

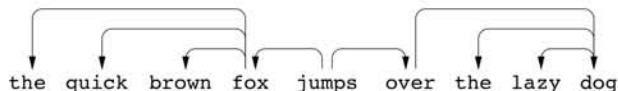

Figure 1: An example parse; arrows are drawn from head words to their dependents. The root word is `jumps`; `brown` is a predependent (adjunct) of `fox`; `dog` is a postdependent (complement) of `over`.

tional models; as a result, grammatical bigram models can be trained efficiently from raw text using an $O(n^3)$ EM algorithm. We present the results of experiments that quantify the representational adequacy of the grammatical bigram model, its ability to generalize from labelled data, and its ability to induce syntactic structure from large amounts of raw text.

## 2   The Grammatical Bigram Model

We first provide a brief introduction to the Dependency Grammar formalism used by the grammatical bigram model; then, we present the probability model and relate it to conventional models; finally, we sketch the EM algorithm for training the model. Details regarding the parsing and learning algorithms can be found in a companion technical report [4].

**Dependency Grammar Formalism.**[1] The primary unit of syntactic structure in dependency grammars is the dependency relationship, or **link**—a binary relation between a pair of words in the sentence. In each link, one word is designated the **head**, and the other is its **dependent**. (Typically, different types of dependency are distinguished, e.g, subject, complement, adjunct, etc.; in our simple model, no such distinction is made.) Dependents that precede their head are called **predependents**, and dependents that follow their heads are called **postdependents**.

A **dependency parse** consists of a set of links that, when viewed as a directed graph over word tokens, form an ordered tree. This implies three important properties:

1. Every word except one (the **root**) is dependent to exactly one head.
2. The links are acyclic; no word is, through a sequence of links, dependent to itself.
3. When drawn as a graph above the sentence, no two dependency relations cross—a property known as **projectivity** or **planarity**.

The planarity constraint ensures that a head word and its (direct or indirect) dependents form a contiguous subsequence of the sentence; this sequence is the head word's **constituent**. See Figure 1 for an example dependency parse.

In order to formalize our dependency grammar model, we will view sentences as sequences of word tokens drawn from some set of word types. Let $V = \{t_1, t_2, \dots, t_M\}$ be our vocabulary of $M$ word types. A sentence with $n$ words is therefore represented as a sequence $S = \langle w_1, w_2, \dots, w_n \rangle$, where each word token $w_i$ is a variable that ranges over $V$. For $1 \le i, j \le n$, we use the notation $(i, j) \in L$ to express that $w_j$ is a dependent of $w_i$ in the parse $L$.

Because it simplifies the structure of our model, we will make the following three assumptions about $S$ and $L$ (without loss of generality): (1) the first word $w_1$ of $S$ is a special symbol ROOT $\in V$; (2) the root of $L$ is $w_1$; and, (3) $w_1$ has only one dependent. These assumptions are merely syntactic sugar: they allow us to treat all words in the true sentence (i.e., $\langle w_2, \ldots, w_n \rangle$) as dependent to one word. (The true root of the sentence is the sole child of $w_1$.)

**Probability Model.** A probabilistic dependency grammar is a probability distribution $P(S, L)$ where $S = \langle w_1, w_2, \ldots, w_n \rangle$ is a sentence, $L$ is a parse of $S$, and the words $w_2, \ldots, w_n$ are random variables ranging over $V$. Of course, $S$ and $L$ exist in high dimensional spaces; therefore, tractable representations of this distribution make use of independence assumptions.

Conventional probabilistic dependency grammar models make use of what may be called the **head word hypothesis**: that a head word is the sole (or primary) determinant of how its constituent combines with other constituents. The head word hypothesis constitutes an independence assumption; it implies that the distribution can be safely factored into a product over constituents:

$$P(S, L) = \prod_{i=1}^{n} P\left(\langle w_j : (i,j) \in L \rangle \text{ is the dependent sequence} \mid w_i \text{ is the head}\right)$$

For example, the probability of a particular sequence can be governed by a fixed set of probabilistic phrase-structure rules, as in [6]; alternatively, the predependent and postdependent subsequences can be modeled separately by Markov chains that are specific to the head word, as in [8].

Consider a much stronger independence assumption: that all the dependents of a head word are independent of one another and their relative order. This is clearly an approximation; in general, there will be strong correlations between the dependents of a head word. More importantly, this assumption prevents the model from representing important argument structure constraints. (For example: many words require dependents, e.g. prepositions; some verbs can have optional objects, whereas others require or forbid them.) However, this assumption relieves the parser of having to maintain internal state for each constituent it constructs, and therefore reduces the computational complexity of parsing and learning.

We can express this independence assumption in the following way: first, we forego modeling the length of the sentence, $n$, since in parsing applications it is always known; then, we expand $P(S, L \mid n)$ into $P(S \mid L)P(L \mid n)$ and choose $P(L \mid n)$ as uniform; finally, we select

$$P(S \mid L) \stackrel{\triangle}{=} \prod_{(i,j) \in L} P(w_j \text{ is a [pre/post]dependent} \mid w_i \text{ is the head})$$

This distribution factors into a product of terms over syntactically related word pairs; therefore, we call this model the "grammatical bigram" model.

The parameters of the model are

$$\gamma_{xy}^{\leftarrow} \stackrel{\triangle}{=} P(\text{predependent is } t_y \mid \text{head is } t_x)$$

$$\gamma_{xy}^{\rightarrow} \stackrel{\triangle}{=} P(\text{postdependent is } t_y \mid \text{head is } t_x)$$

We can make the parameterization explicit by introducing the indicator variable $w_i^x$, whose value is 1 if $w_i = t_x$ and 0 otherwise. Then we can express $P(S \mid L)$ as

$$P(S \mid L) \stackrel{\triangle}{=} \prod_{\substack{(i,j) \in L \\ j < i}} \prod_{x=1}^{M} \prod_{y=1}^{M} \left[\gamma_{xy}^{\leftarrow}\right]^{w_i^x w_j^y} \prod_{\substack{(i,j) \in L \\ i < j}} \prod_{x=1}^{M} \prod_{y=1}^{M} \left[\gamma_{xy}^{\rightarrow}\right]^{w_i^x w_j^y}$$

**Parsing.** Parsing a sentence $S$ consists of computing

$$L^* \triangleq \arg\max_L P(L \mid S, n) = \arg\max_L P(L, S \mid n) = \arg\max_L P(S \mid L)$$

Yuret has shown that there are exponentially many parses of a sentence with $n$ words [9], so exhaustive search for $L^*$ is intractable. Fortunately, our grammar model falls into the class of "Bilexical Grammars", for which efficient parsing algorithms have been developed. Our parsing algorithm (described in the tech report [4]) is derived from Eisner's span-based chart-parsing algorithm [5], and can find $L^*$ in $O(n^3)$ time.

**Learning.** Suppose we have a labelled data set

$$\mathcal{D} = \{(S_1, L_1, n_1), (S_2, L_2, n_2), \dots, (S_N, L_N, n_N)\}$$

where $S_k = (w_{1,k}, w_{2,k}, \dots, w_{n_k,k})$ and $L_k$ is a parse over $S_k$. The maximum likelihood values for our parameters given the training data are

$$\widehat{\overrightarrow{\gamma}_{xy}} = \frac{\sum_{k=1}^N \sum_{1 \leq i < j \leq n_k} e_{ij}^k w_{i,k}^x w_{j,k}^y}{\sum_{k=1}^N \sum_{1 \leq i < j \leq n} e_{ij}^k w_{i,k}^x} \qquad \widehat{\overleftarrow{\gamma}_{xy}} = \frac{\sum_{k=1}^N \sum_{1 \leq j < i \leq n_k} e_{ij}^k w_{i,k}^x w_{j,k}^y}{\sum_{k=1}^N \sum_{1 \leq j < i \leq n_k} e_{ij}^k w_{i,k}^x}$$

where the indicator variable $e_{ij}^k$ is equal to 1 if $(i, j) \in L_k$ and 0 otherwise. As one would expect, the maximum-likelihood value of $\overleftarrow{\gamma}_{xy}$ (resp. $\overrightarrow{\gamma}_{xy}$) is simply the fraction of $t_x$'s predependents (resp. postdependents) that were $t_y$.

In the unsupervised acquisition problem, our data set has no parses; our approach is to treat the $L_k$ as hidden variables and to employ the EM algorithm to learn (locally) optimal values of the parameters $\gamma$. As we have shown above, the $e_{ij}^k$ are sufficient statistics for our model; the companion tech report [4] gives an adaptation of the Inside-Outside algorithm which computes their conditional expectation in $O(n^3)$ time. This algorithm effectively examines every possible parse of every sentence in the training set and calculates the expected number of times each pair of words was related syntactically.

# 3   Evaluation

This section presents three experiments that attempt to quantify the representational adequacy and learnability of grammatical bigram models.

**Corpora.** Our experiments make use of two corpora; one is labelled with parses, and the other is not. The labelled corpus was generated automatically from the phrase-structure trees in the Wall Street Journal portion of the Penn Treebank-III [10].[2] The resultant corpus, which we call $\mathcal{L}$, consists of 49,207 sentences (1,037,374 word tokens). This corpus is split into two pieces: 90% of the sentences comprise corpus $\mathcal{L}_{\text{train}}$ (44,286 sentences, 934,659 word tokens), and the remaining 10% comprise $\mathcal{L}_{\text{test}}$ (4,921 sentences, 102,715 word tokens).

The unlabelled corpus consists of the 1987–1992 Wall Street Journal articles in the TREC Text Research Collection Volumes 1 and 2. These articles were segmented on sentence boundaries using the technique of [11], and the sentences were post-processed to have a format similar to corpus $\mathcal{L}$. The resultant corpus consists of 3,347,516 sentences (66,777,856 word tokens). We will call this corpus $\mathcal{U}$.

The model's vocabulary is the same for all experiments; it consists of the 10,000 most frequent word types in corpus $\mathcal{U}$; this vocabulary covers 94.0% of word instances in corpus $\mathcal{U}$ and 93.9% of word instances in corpus $\mathcal{L}$. Words encountered during testing and training that are outside the vocabulary are mapped to the <unk> type.

**Performance metric.** The performance metric we report is the **link precision** of the grammatical bigram model: the fraction of links hypothesized by the model that are present in the test corpus $\mathcal{L}_{\text{test}}$. (In a scenario where the model is not required to output a complete parse, e.g., a shallow parsing task, we could similarly define a notion of link recall; but in our current setting, these metrics are identical.) Link precision is measured without regard for link orientation; this amounts to ignoring the model's choice of root, since this choice induces a directionality on all of the edges.

**Experiments.** We report on the results of three experiments:

I. *Retention.* This experiment represents a best-case scenario: the model is trained on corpora $\mathcal{L}_{\text{train}}$ and $\mathcal{L}_{\text{test}}$ and then tested on $\mathcal{L}_{\text{test}}$. The model's link precision in this setting is 80.6%.

II. *Generalization.* In this experiment, we measure the model's ability to generalize from labelled data. The model is trained on $\mathcal{L}_{\text{train}}$ and then tested on $\mathcal{L}_{\text{test}}$. The model's link precision in this setting is 61.8%.

III. *Induction.* In this experiment, we measure the model's ability to induce grammatical structure from unlabelled data. The model is trained on $\mathcal{U}$ and then tested on $\mathcal{L}_{\text{test}}$. The model's link precision in this setting is 39.7%.

**Analysis.** The results of Experiment I give some measure of the grammatical bigram model's representational adequacy. A model that memorizes every parse would perform perfectly in this setting, but the grammatical bigram model is only able to recover four out of every five links. To see why, we can examine an example parse. Figure 2 shows how the models trained in Experiments I, II, and III parse the same test sentence. In the top parse, syndrome is incorrectly selected as a postdependent of the first on token rather than the second. This error can be attributed directly to the grammatical bigram independence assumption: because argument structure is not modeled, there is no reason to prefer the correct parse, in which both on tokens have a single dependent, over the chosen parse, in which the first has two dependents and the second has none.[3]

Experiment II measures the generalization ability of the grammatical bigram model; in this setting, the model can recover three out of every five links. To see why the performance drops so drastically, we again turn to an example parse: the middle parse in Figure 2. Because the forces → on link was never observed in the training data, served has been made the head of both on tokens; ironically, this corrects the error made in the top parse because the planarity constraint rules out the incorrect link from the first on token to syndrome. Another error in the middle parse is a failure to select several as a predependent of forces; this error also arises because the combination never occurs in the training data. Thus, we can attribute this drop in performance to sparseness in the training data.

We can compare the grammatical bigram model's parsing performance with the results reported by Eisner [8]. In that investigation, several different probability models are ascribed to the simple dependency grammar described above and

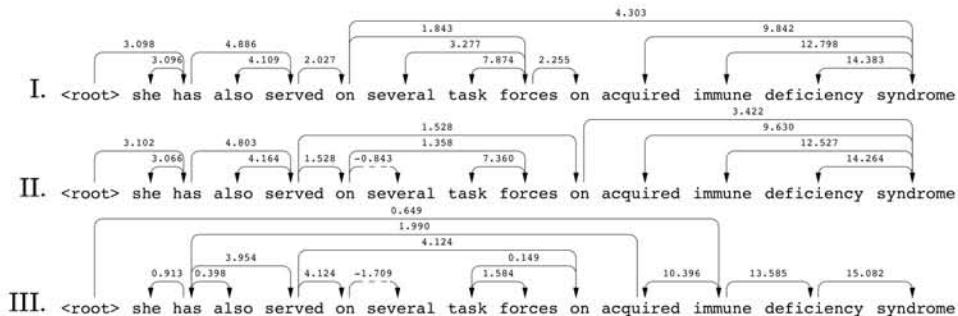

Figure 2: The same test sentence, parsed by the models trained in each of the three experiments. Links are labelled with $-\log_2 \gamma_{xy} / \sum_{x=1}^{M} \gamma_{xy}$, the mutual information of the linked words; dotted edges are default attachments.

are compared on a task similar to Experiment II.[4] Eisner reports that the best-performing dependency grammar model (Model D) achieves a (direction-sensitive) link precision of 90.0%, and the Collins parser [6] achieves a (direction-sensitive) link precision of 92.6%. The superior performance of these models can be attributed to two factors: first, they include sophisticated models of argument structure; and second, they both make use of part-of-speech taggers, and can "back-off" to non-lexical distributions when statistics are not available.

Finally, Experiment III shows that when trained on unlabelled data, the grammatical bigram model is able to recover two out of every five links. This performance is rather poor, and is only slightly better than chance; a model that chooses parses uniformly at random achieves 31.3% precision on $\mathcal{L}_{\text{test}}$. To get an intuition for why this performance is so poor, we can examine the last parse, which was induced from unlabelled data. Because Wall Street Journal articles often report corporate news, the frequent co-occurrence of `has` → `acquired` has led to a parse consistent with the interpretation that the subject `she` suffers from AIDS, rather than serving on a task force to study it. We also see that a flat parse structure has been selected for `acquired immune deficiency syndrome`; this is because while this particular noun phrase occurs in the training data, its constituent nouns do not occur independently with any frequency, and so their relative co-occurrence frequencies cannot be assessed.

## 4 Discussion

**Future work.** As one would expect, our experiments indicate that the parsing performance of the grammatical bigram model is not as good as that of state-of-the-art parsers; however, its performance in Experiment II suggests that it may be useful in domains where partial accuracy is valuable and large amounts of unlabelled data are available. However, to realize that potential, the model must be improved so that its performance in Experiment III is closer to that of Experiment II.

To that end, we can see two obvious avenues of improvement. The first involves increasing the model's capacity for generalization and preventing overfitting. The

model presented in this paper is sensitive only to pairwise relationships to words; however, it could make good use of the fact that words can have similar syntactic behavior. We are currently investigating whether word clustering techniques can improve performance in supervised and unsupervised learning. Another way to improve the model is to directly address the primary source of parsing error: the lack of argument structure modeling. We are also investigating approximation algorithms that reintroduce argument structure constraints without making the computational complexity unmanageable.

**Related work.** A recent proposal by Yuret presents a "lexical attraction" model with similarities to the grammatical bigram model [9]; however, unlike the present proposal, that model is trained using a heuristic algorithm. The grammatical bigram model also bears resemblance to several proposals to extend finite-state methods to model long-distance dependencies (e.g., [12, 13]), although these models are not based upon an underlying theory of syntax.

## Footnotes

[1]The Dependency Grammar formalism described here (which is the same used in [5, 6]) is impoverished compared to the sophisticated models used in Linguistics; refer to [7] for a comprehensive treatment of English syntax in a dependency framework.

[2]This involved selecting a head word for each constituent, for which the head-word extraction heuristics described in [6] were employed. Additionally, punctuation was removed, all words were down-cased, and all numbers were mapped to a special <#> symbol.

[3]Although the model's parse of acquired immune deficiency syndrome agrees with the labelled corpus, this particular parse reflects a failure of the head-word extraction heuristics; acquired and immune should be predependents of deficiency, and deficiency should be a predependent of syndrome.

[4]The labelled corpus used in that investigation is also based upon a transformed version of Treebank-III, but the head-word extraction heuristics were slightly different, and sentences with conjunctions were completely eliminated. However, the setup is sufficiently similar that we think the comparison we draw is informative.

# References

[1] K. Lari and S. J. Young. The estimation of stochastic context-free grammars using the Inside-Outside algorithm. *Computer Speech and Language*, 4:35–56, 1990.

[2] John Lafferty, Daniel Sleator, and Davy Temperley. Grammatical trigrams: A probabilistic model of link grammar. In *Proceedings of the AAAI Conference on Probabilistic Approaches to Natural Language*, October 1992.

[3] Yves Schabes. Stochastic lexicalized tree-adjoining grammars. In *Proceedings of the Fourteenth International Conference on Computational Linguistics*, pages 426–432, Nantes, France, 1992.

[4] Mark A. Paskin. Cubic-time parsing and learning algorithms for grammatical bigram models. Technical Report CSD-01-1148, University of California, Berkeley, 2001.

[5] Jason Eisner. Bilexical grammars and their cubic-time parsing algorithms. In Harry Bunt and Anton Nijholt, editors, *Advances in Probabilistic and Other Parsing Technologies*, chapter 1. Kluwer Academic Publishers, October 2000.

[6] Michael Collins. *Head-driven Statistical Models for Natural Language Parsing*. PhD thesis, University of Pennsylvania, Philadelphia, Pennsylvania, 1999.

[7] Richard A. Hudson. *English Word Grammar*. B. Blackwell, Oxford, UK, 1990.

[8] Jason M. Eisner. An empirical comparison of probability models for dependency grammars. Technical Report ICRS-96-11, CIS Department, University of Pennsylvania, 220 S. 33$^{rd}$ St. Philadelphia, PA 19104–6389, 1996.

[9] Deniz Yuret. *Discovery of Linguistic Relations Using Lexical Attraction*. PhD thesis, Massachusetts Institute of Technology, May 1998.

[10] M. Marcus, B. Santorini, and M. Marcinkiewicz. Building a large annotated corpus of english: The penn treebank. *Computational Linguistics*, 19:313–330, 1993.

[11] Jeffrey C. Reynar and Adwait Ratnaparkhi. A maximum entropy approach to identifying sentence boundaries. In *Proceedings of the Fifth Conference on Applied Natural Language Processing*, Washington, D.C., March 31 – April 3 1997.

[12] S. Della Pietra, V. Della Pietra, J. Gillett, J. Lafferty, H. Printz, and L. Ures. Inference and estimation of a long-range trigram model. In *Proceedings of the Second International Colloquium on Grammatical Inference and Applications*, number 862 in Lecture Notes in Artificial Intelligence, pages 78–92. Springer-Verlag, 1994.

[13] Ronald Rosenfeld. *Adaptive Statistical Language Modeling: A Maximum Entropy Approach*. PhD thesis, Carnegie Mellon University, 1994.
